# Nonnegative dictionary learning in the exponential noise model for adaptive music signal representation

**Onur Dikmen**
CNRS LTCI; Télécom ParisTech
75014, Paris, France
dikmen@telecom-paristech.fr

**Cédric Févotte**
CNRS LTCI; Télécom ParisTech
75014, Paris, France
fevotte@telecom-paristech.fr

## Abstract

In this paper we describe a maximum likelihood approach for dictionary learning in the multiplicative exponential noise model. This model is prevalent in audio signal processing where it underlies a generative composite model of the power spectrogram. Maximum joint likelihood estimation of the dictionary and expansion coefficients leads to a nonnegative matrix factorization problem where the Itakura-Saito divergence is used. The optimality of this approach is in question because the number of parameters (which include the expansion coefficients) grows with the number of observations. In this paper we describe a variational procedure for optimization of the marginal likelihood, i.e., the likelihood of the dictionary where the activation coefficients have been integrated out (given a specific prior). We compare the output of both maximum joint likelihood estimation (i.e., standard Itakura-Saito NMF) and maximum marginal likelihood estimation (MMLE) on real and synthetical datasets. The MMLE approach is shown to embed automatic model order selection, akin to automatic relevance determination.

## 1  Introduction

In this paper we address the task of nonnegative dictionary learning described by

$$V \approx WH, \tag{1}$$

where $V$, $W$, $H$ are nonnegative matrices of dimensions $F \times N$, $F \times K$ and $K \times N$, respectively. $V$ is the data matrix, where each column $v_n$ is a data point, $W$ is the dictionary matrix, with columns $\{w_k\}$ acting as "patterns" or "explanatory variables" reprensentative of the data, and $H$ is the activation matrix, with columns $\{h_n\}$. For example, in this paper we will be interested in music data such that $V$ is time-frequency spectrogram matrix and $W$ is a collection of spectral signatures of latent elementary audio components. The most common approach to nonnegative dictionary learning is nonnegative matrix factorization (NMF) [1] which consists in retrieving the factorization (1) by solving

$$\min_{W,H} D(V|WH) \overset{\text{def}}{=} \sum_{fn} d(v_{fn}|[WH]_{fn}) \quad \text{s.t.} \quad W, H \geq 0 , \tag{2}$$

where $d(x|y)$ is a measure of fit between nonnegative scalars, $v_{fn}$ are the entries of $V$, and $A \geq 0$ expresses nonnegativity of the entries of matrix $A$. The cost function $D(V|WH)$ is often a likelihood function $-\log p(V|W, H)$ in disguise, e.g., the Euclidean distance underlies additive Gaussian noise, the Kullback-Leibler (KL) divergence underlies Poissonian noise, while the Itakura-Saito (IS) divergence underlies multiplicative exponential noise [2]. The latter noise model will be central to this work because it underlies a suitable generative model of the power spectrogram, as shown in [3] and later recalled.

A criticism about NMF is that little can be said about the asymptotical optimality of the learnt dictionary $W$. Indeed, because $W$ is estimated jointly with $H$, the total number of parameters $FK + KN$ grows with the number of data points $N$. As such, this paper instead addresses optimization of the likelihood in the marginal model described by

$$p(V|W) = \int_H p(V|W, H)p(H)dH, \tag{3}$$

where $H$ is treated as a random latent variable with prior $p(H)$. The evaluation and optimization of the marginal likelihood is not trivial in general, and this paper is precisely devoted to these tasks in the multiplicative exponential noise model.

The maximum marginal likelihood estimation approach we seek here is related to IS-NMF in such a way that Latent Dirichlet Allocation (LDA) [4] is related to Latent Semantic Indexing (pLSI) [5]. LDA and pLSI are two estimators in the same model, but LDA seeks estimation of the topic distributions in the marginal model, from which the topic weights describing each document have been integrated out. In contrast, pLSI (which is essentially equivalent to KL-NMF as shown in [6]) performs maximum *joint* likelihood estimation (MJLE) for the topics and weights. Blei *et al.* [4] show the better performance of LDA with respect to (w.r.t) pLSI. Welling *et al.* [7] also report similar results with a discussion, stating that deterministic latent variable models assign zero probability to input configurations that do not appear in the training set. A similar approach is Discrete Component Analysis (DCA) [8] which considers maximum marginal a posteriori estimation in the Gamma-Poisson (GaP) model [9], see also [10] for the maximum marginal likelihood estimation on the same model. In this paper, we will follow the same objective for the multiplicative exponential noise model.

We will describe a variational algorithm for the evaluation and optimization of (3); note that the algorithm exploits specificities of the model and is not a mere adaptation of LDA or DCA to an alternative setting. We will consider a nonnegative Generalized inverse-Gaussian (GIG) distribution as a prior for $H$, a flexible distribution which takes the Gamma and inverse-Gamma as special cases. As will be detailed later, this work relates to recent work by Hoffman *et al.* [11], which considers full Bayesian integration of $W$ and $H$ (both assumed random) in the exponential noise model, in a nonparametric setting allowing for model order selection. We will show that our more simple maximum likelihood approach inherently performs model selection as well by automatically pruning "irrelevant" dictionary elements. Applied to a short well structured piano sequence, our approach is shown to capture the correct number of components, corresponding to the expected note spectra, and outperforms the nonparametric Bayesian approach of [11].

The paper is organized as follows. Section 2 introduces the multiplicative exponential noise model with the prior distribution for the expansion coefficients $p(H)$. Sections 3 and 4 describe the MJLE and MMLE approaches, respectively. Section 5 reports results on synthetical and real audio data. Section 6 concludes.

## 2 Model

The generative model assumed in this paper is

$$v_{fn} = \hat{v}_{fn} . \epsilon_{fn} , \tag{4}$$

where $\hat{v}_{fn} = \sum_k w_{fk} h_{kn}$ and $\epsilon_{fn}$ is a nonnegative multiplicative noise with exponential distribution $\epsilon_{fn} \sim \exp(-\epsilon_{fn})$. In other words, and under independence assumptions, the likelihood function is

$$p(V|W, H) = \prod_{fn} (1/\hat{v}_{fn}) \exp(-v_{fn}/\hat{v}_{fn}) . \tag{5}$$

When $V$ is a power spectrogram matrix such that $v_{fn} = |x_{fn}|^2$ and $\{x_{fn}\}$ are the complex-valued short-time Fourier transform (STFT) coefficients of some signal data, where $f$ typically acts as a frequency index and $n$ acts as a time-frame index, it was shown in [3] that an equivalent generative model of $v_{fn}$ is

$$x_{fn} = \sum_k c_{fkn}, \quad c_{fkn} \sim \mathcal{N}_c(0, w_{fk} h_{kn}) , \tag{6}$$

where $\mathcal{N}_c$ refers to the circular complex Gaussian distribution.[1] In other words, the exponential multiplicative noise model underlies a generative composite model of the STFT. The complex-valued matrix $\{c_{fkn}\}_{fn}$, referred to as $k^{th}$ component, is characterized by a spectral signature $w_k$, amplitude-modulated in time by the frame-dependent coefficient $h_{kn}$, which accounts for nonstationarity. In analogy with LDA or DCA, if our data consisted of word counts, with $f$ indexing words and $n$ indexing documents, then the columns of $W$ would describe topics and $c_{fkn}$ would denote the number of occurrences of word $f$ stemming from topic $k$ in document $n$.

In our setting $W$ is considered a free deterministic parameter to be estimated by maximum likelihood. In contrast, $H$ is treated as a nonnegative random latent variable over which we will integrate. It is assigned a GIG prior, such that

$$h_{kn} \sim \mathcal{GIG}(\alpha_k, \beta_k, \gamma_k) \,, \tag{7}$$

with

$$\mathcal{GIG}(x|\alpha, \beta, \gamma) = \frac{(\beta/\gamma)^{\alpha/2}}{2\mathcal{K}_\alpha(2\sqrt{\beta\gamma})} x^{\alpha-1} \exp - \left(\beta x + \frac{\gamma}{x}\right) \,, \tag{8}$$

where $\mathcal{K}$ is a modified Bessel function of the second kind and $x$, $\beta$ and $\gamma$ are nonnegative scalars. The GIG distribution unifies the Gamma ($\alpha > 0$, $\gamma = 0$) and inverse-Gamma ($\alpha < 0$, $\beta = 0$) distributions. Its sufficient statistics are $x$, $1/x$ and $\log x$, and in particular we have

$$\langle x \rangle = \frac{\mathcal{K}_{\alpha+1}(2\sqrt{\beta\gamma})}{\mathcal{K}_\alpha(2\sqrt{\beta\gamma})} \sqrt{\frac{\gamma}{\beta}}, \qquad \left\langle \frac{1}{x} \right\rangle = \frac{\mathcal{K}_{\alpha-1}(2\sqrt{\beta\gamma})}{\mathcal{K}_\alpha(2\sqrt{\beta\gamma})} \sqrt{\frac{\beta}{\gamma}}, \tag{9}$$

where $\langle x \rangle$ denotes expectation. Although all derivations and the implementations are done for the general case, in practice we will only consider the special case of Gamma distribution for simplicity. In such case, $\beta$ parameter merely acts as a scale parameter, which we fix so as to solve the scale ambiguity between the columns of $W$ and the rows of $H$. We will also assume the shape parameters $\{\alpha_k\}$ fixed to arbitrary values (typically, $\alpha_k = 1$, which corresponds to the exponential distribution). Given the generative model specified by equations (4) and (7) we now describe two estimators for $W$.

## 3 Maximum joint likelihood estimation

### 3.1 Estimator

The joint (penalized) log-likelihood likelihood of $W$ and $H$ is defined by

$$C_{\mathrm{JL}}(W, H) \overset{\text{def}}{=} \log p(V|W, H) + \log p(H) \tag{10}$$

$$= -D_{\mathrm{IS}}(V|WH) - \sum_{kn}(1 - \alpha_k)\log h_{kn} + \beta_k h_{kn} + \gamma_k/h_{kn} + \mathrm{cst} \,, \tag{11}$$

where $D_{\mathrm{IS}}(V|WH)$ is defined as in equation (2) with $d_{\mathrm{IS}}(x|y) = x/y - \log(x/y) - 1$ (Itakura-Saito divergence) and "cst" denotes terms constant w.r.t $W$ and $H$. The subscript JL stands for joint likelihood, and the estimation of $W$ by maximization of $C_{\mathrm{JL}}(W, H)$ will be referred to as *maximum joint likelihood estimation* (MJLE).

### 3.2 MM algorithm for MJLE

We describe an iterative algorithm which sequentially updates $W$ given $H$ and $H$ given $W$. Each of the two steps can be achieved in a *minorization-maximization* (MM) setting [12], where the original problem is replaced by the iterative optimization of an easier-to-optimize auxiliary function. We first describe the update of $H$, from which the update of $W$ will be easily deduced. Given $W$, our task consists in maximizing $C(H) = -D_{\mathrm{IS}}(V|WH) - L(H)$, where $L(H) = \sum_{kn}(1 - \alpha_k)\log h_{kn} + \beta_k h_{kn} + \gamma_k/h_{kn}$. Using Jensen's inequality to majorize the convex part of $D_{\mathrm{IS}}(V|WH)$ (terms in

$v_{fn}/\hat{v}_{fn}$) and first order Taylor approximation to majorize its concave part (terms in $\log \hat{v}_{fn}$), as in [13], the functional

$$G(H, \tilde{H}) = - \left( \sum_k p_{kn}/h_{kn} + q_{kn}h_{kn} \right) - L(H) + \text{cst}, \qquad (12)$$

where $p_{kn} = \tilde{h}_{kn}^2 \sum_f w_{fk}v_{fn}/\tilde{v}_{fn}^2$, $q_{kn} = \sum_f w_{fk}/\tilde{v}_{fn}$, $\tilde{v}_{fn} = [W\tilde{H}]_{fn}$, can be shown to be a tight lower bound of $C(H)$, i.e., $G(H, \tilde{H}) \leq C(H)$ and $G(\tilde{H}, \tilde{H}) = C(\tilde{H})$. Its iterative maximization w.r.t $H$, where $\tilde{H} = H^{(i)}$ acts as the current iterate at iteration $i$, produces an ascent algorithm, such that $C(H^{(i+1)}) \geq C(H^{(i)})$. The update is easily shown to amount to solving an order 2 polynomial with a single positive root given by

$$h_{kn} = \frac{(\alpha_k - 1) + \sqrt{(\alpha_k - 1)^2 + 4(p_{kn} + \gamma_k)(q_{kn} + \beta_k)}}{2(q_{kn} + \beta_k)}. \qquad (13)$$

The update preserves nonnegativity given positive initialization. By exchangeability of $W$ and $H$ when the data is transposed ($V^T = H^T W^T$), and dropping the penalty term ($\alpha_k = 1$, $\beta_k = 0$, $\gamma_k = 0$), the update of $W$ is given by the multiplicative update

$$w_{fk} = \tilde{w}_{fk} \sqrt{\frac{\sum_n h_{kn}v_{fn}/\tilde{v}_{fn}^2}{\sum_n h_{kn}/\tilde{v}_{fn}}}, \qquad (14)$$

which is known from [13].

## 4 Maximum marginal likelihood estimation

### 4.1 Estimator

We define the marginal log-likelihood objective function as

$$C_{\mathrm{ML}}(W) \overset{\text{def}}{=} \log \int p(V|W, H)p(H)\, dH\,. \qquad (15)$$

The subscript ML stands for marginal likelihood, and the estimation of $W$ by maximization of $C_{\mathrm{ML}}(W)$ will be referred to as *maximum marginal likelihood estimation* (MMLE). Note that in Bayesian estimation the term *marginal likelihood* is sometimes used as a synonym for the *model evidence*, which is the likelihood of data given the model, i.e., where all random parameters (including $W$) have been marginalized. This is not the case here where $W$ is treated as a deterministic parameter and marginal likelihood only refers to the likelihood of $W$, where $H$ has been integrated out. The integral in equation (15) is intractable given our model. In the next section we resort to a variational Bayes procedure for the evaluation and maximization of $C_{\mathrm{ML}}(W)$.

### 4.2 Variational algorithm for MMLE

In the following we propose an iterative lower bound evaluation/maximization procedure for approximate maximization of $C_{\mathrm{ML}}(W)$. We will construct a bound $B(W, \tilde{W})$ such that $\forall (W, \tilde{W}), C_{\mathrm{ML}}(W) \geq B(W, \tilde{W})$, where $\tilde{W}$ acts as the current iterate and $W$ acts as the free parameter over which the bound is maximized. The maximization is approximate in that the bound will only satisfy $B(\tilde{W}, \tilde{W}) \approx C_{\mathrm{ML}}(\tilde{W})$, i.e., is loosely tight in the current update $\tilde{W}$, which fails to ensure ascent of the objective function like in the MM setting of Section 3.2.

We propose to construct the bound from a variational Bayes perspective [14]. The following inequality holds for any distribution function $q(H)$

$$C_{\mathrm{ML}}(W) \geq \langle \log p(V|W, H) \rangle_q + \langle \log p(H) \rangle_q - \langle \log q(H) \rangle_q \overset{\text{def}}{=} B_q^{\mathrm{vb}}(W)\,. \qquad (16)$$

The inequality becomes an equality when $q(H) = p(H|V, W)$; when the latter is available in close form, the EM algorithm consists in using $\tilde{q}(H) = p(H|V, \tilde{W})$ and maximize $B_{\tilde{q}}^{\mathrm{vb}}(W)$ w.r.t $W$, and iterate. The true posterior of $H$ being intractable in our case, we take $q(H)$ to be a factorized,

parametric distribution $q_\theta(H)$, whose parameter $\theta$ is updated so as to tighten $B_q^{\text{vb}}(\tilde{W})$ to $C(\tilde{W})$. Like in [11], we choose $q_\theta(H)$ to be in the same family as the prior, such that

$$q_\theta(H) = \prod_{kn} \mathcal{GIG}(\bar{\alpha}_{kn}, \bar{\beta}_{kn}, \bar{\gamma}_{kn}). \tag{17}$$

The first term of $B_q^{\text{vb}}(W)$ essentially involves the expectation of $-D_{\text{IS}}(V|WH)$ w.r.t to the variational distribution $q_\theta(H)$. The product $WH$ introduces some coupling of the coefficients of $H$ (via the sum $\sum_k w_{fk} h_{kn}$) which makes the integration difficult. Following [11] and similar to Section 3.2, we propose to lower bound this term using Jensen's and Taylor's type inequalities to majorize the convex and concave parts of $-D_{\text{IS}}(V|WH)$. The contributions of the elements of $H$ become decoupled w.r.t to $k$, which allows for evaluation and maximization of the bound. This leads to

$$\langle \log p(V|H,W) \rangle_q \geq -\sum_{fn} \left( \sum_k \phi_{fkn}^2 \frac{v_{fn}}{w_{fk}} \left\langle \frac{1}{h_{kn}} \right\rangle_q \right) + \left( \log \psi_{fn} + \frac{1}{\psi_{fn}} \sum_k w_{fk} \langle h_{kn} \rangle_q - 1 \right), \tag{18}$$

where $\{\psi_{fn}\}$ and $\{\phi_{fkn}\}$ are nonnegative free parameters such that $\sum_k \phi_{fkn} = 1$. We define $B_{\theta,\phi,\psi}(W)$ as $B_q^{\text{vb}}(W)$ but where the expectation of the joint log-likelihood is replaced by its lower bound given right side of equation (18). From there, our algorithm is a two-step procedure consisting in 1) computing $\tilde{\theta}, \tilde{\phi}, \tilde{\psi}$ so as to tighten $B_{\theta,\phi,\psi}(\tilde{W})$ to $C_{\text{ML}}(\tilde{W})$, and 2) maximizing $B_{\tilde{\theta},\tilde{\phi},\tilde{\psi}}(W)$ w.r.t $W$. The corresponding updates are given next. Note that evaluation of the bound only involves expectations of $h_{kn}$ and $1/h_{kn}$ w.r.t to the GIG distribution, which is readily given by equation (9).

**Step 1: Tightening the bound**   Given current dictionary update $\tilde{W}$, run the following fixed-point equations.

$$\phi_{fkn} = \frac{\tilde{w}_{fk}/\langle 1/h_{kn} \rangle_q}{\sum_j \tilde{w}_{fj}/\langle 1/h_{jn} \rangle_q}, \qquad \psi_{fn} = \sum_j \tilde{w}_{fj} \langle h_{jn} \rangle_q$$

$$\bar{\alpha}_{kn} = \alpha_k, \qquad \bar{\beta}_{kn} = \beta_k + \sum_f \frac{\tilde{w}_{fk}}{\psi_{fn}}, \qquad \bar{\gamma}_{kn} = \gamma_k + \sum_f \frac{v_{fn} \phi_{fkn}^2}{\tilde{w}_{fk}}.$$

**Step 2: Optimizing the bound**   Given the variational distribution $\tilde{q} = q_{\tilde{\theta}}$ from previous step, update $W$ as

$$w_{fk} = \tilde{w}_{fk} \sqrt{\frac{\sum_n v_{fn} \left[ \sum_j \tilde{w}_{fj} \langle 1/h_{jn} \rangle_{\tilde{q}}^{-1} \right]^{-2} \langle 1/h_{kn} \rangle_{\tilde{q}}^{-1}}{\sum_n \left[ \sum_j \tilde{w}_{fj} \langle h_{jn} \rangle_{\tilde{q}} \right]^{-1} \langle h_{kn} \rangle_{\tilde{q}}}}. \tag{19}$$

The VB update has a similar form to the MM update of equation (14) but the contributions of $H$ are replaced by expected values w.r.t the variational distribution.

### 4.3   Relation to other works

A variational algorithm using the activation matrix $H$ and the latent components $C = \{c_{fkn}\}$ as hidden data can easily be devised, as sketched in [2]. Including $C$ in the variational distribution also allows to decouple the contributions of the activation coefficients w.r.t to $k$ but leads from our experience to a looser bound, a finding also reported in [11]. In a fully Bayesian setting, Hoffman *et al.* [11] assume Gamma priors for both $W$ and $H$. The model is such that $\hat{v}_{fn} = \sum_k \lambda_k w_{fk} h_{kn}$, where $\lambda_k$ acts as a component weight parameter. The number of components is potentially infinite but, in a nonparametric setting, the prior for $\lambda_k$ favors a finite number of active components. Posterior inference of the parameters $W$, $H$, $\{\lambda_k\}$ is achieved in a variational setting similar to Section 4.2, by maximizing a lower bound on $p(V)$. In contrast to this method, our approach does not require to specify a prior for $W$, leads to simple updates for $W$ that are directly comparable to IS-NMF and experiments will reveal that our approach embeds model order selection as well, by automatically pruning unnecessary columns of $W$, without resorting to the nonparametric framework.

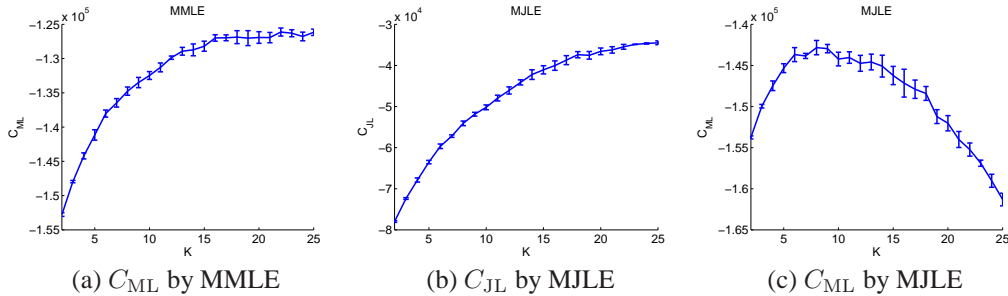

(a) $C_{\mathrm{ML}}$ by MMLE · (b) $C_{\mathrm{JL}}$ by MJLE · (c) $C_{\mathrm{ML}}$ by MJLE

Figure 1: Marginal likelihood $C_{\mathrm{ML}}$ (a) and joint likelihood $C_{\mathrm{JL}}$ (b) versus number of components $K$. $C_{\mathrm{ML}}$ values corresponding to dictionaries estimated by $C_{\mathrm{JL}}$ maximization (c).

## 5  Experiments

In this section, we study the performances of MJLE and MMLE methods on both synthetical and real-world datasets.[2] The prior hyperparameters are fixed to $\alpha_k = 1$, $\gamma_k = 0$ (exponential distribution) and $\beta_k = 1$, i.e., $h_{kn} \sim \exp(-h_{kn})$. We used 5000 algorithm iterations and nonnegative random initializations in all cases. In order to minimize the odds of getting stuck in local optima, we adapted the deterministic annealing method proposed in [15] for MMLE. Deterministic annealing is applied by multiplying the entropy term $-\langle \log q(H) \rangle$ in the lower bound in (16) by $1/\eta^{(i)}$. The initial $\eta^{(0)}$ is chosen in $(0,1)$ and increased through iterations. In our experiments, we set $\eta^{(0)} = 0.6$ and updated it with the rule $\eta^{(i+1)} = \min(1, 1.005\eta^{(i)})$.

### 5.1  Swimmer dataset

First, we consider the synthetical `Swimmer` dataset [16], for which the ground truth of the dictionary is available. The dataset is composed of 256 images of size $32 \times 32$, representing a swimmer built of an invariant torso and 4 limbs. Each of the 4 limbs can be in one of 4 positions and the dataset is formed of all combinations. Hence, the ground truth dictionary corresponds to the collection of individual limb positions. As explained in [16] the torso is an unidentifiable component that can be paired with any of the limbs, or even split among the limbs. In our experiments, we mapped the values in the dataset onto the range $[1, 100]$ and multiplied with exponential noise, see some samples in Fig. 2 (a).

We ran the MM and VB algorithms (for MJLE and MMLE, respectively) for $K = 1 \ldots 20$ and the joint and marginal log-likelihood end values (after the 5000 iterations) are displayed in Fig. 1. The marginal log-likelihood is here approximated by its lower bound, as described in Section 4.2. In Fig. 1(a) and (b) the respective objective criteria ($C_{\mathrm{ML}}$ and $C_{\mathrm{JL}}$) maximized by MMLE and MJLE are shown. The increase of $C_{\mathrm{ML}}$ stops after $K = 16$, whereas $C_{\mathrm{JL}}$ continues to increase as $K$ gets larger. Fig. 1 (c) displays the corresponding marginal likelihood values, $C_{\mathrm{ML}}$, of the dictionaries obtained by MJLE in Fig. 1 (b); this figure empirically shows that maximizing the joint likelihood does not necessarily imply maximization of the marginal likelihood. These figures display the mean and standard deviation values obtained from 7 experiments.

The likelihood values increase with the number of components, as expected from nested models. However, the marginal likelihood stagnates after $K = 16$. Manual inspection reveals that passed this value of $K$, the extra columns of $W$ are pruned to zero, leaving the criterion unchanged. Hence, MMLE appears to embed automatic order selection, similar to automatic relevance determination [17, 18]. The dictionaries learnt from MJLE and MMLE with $K = 20$ components are shown in Fig. 2 (b) and (c). As can be seen from Fig. 2 (b), MJLE produces spurious or duplicated components. In contrast, the ground truth is well recovered with MMLE.

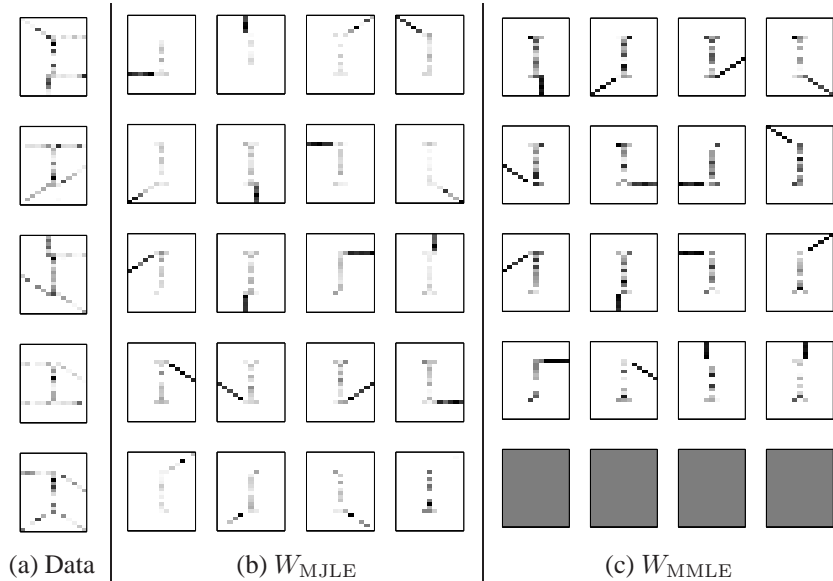

| (a) Data | (b) $W_{\text{MJLE}}$ | (c) $W_{\text{MMLE}}$ |

Figure 2: Data samples and dictionaries learnt on the swimmer dataset with $K = 20$.

## 5.2 A piano excerpt

In this section, we consider the piano data used in [3]. It is a toy audio sequence recorded in real conditions, consisting of four notes played all together in the first measure and in all possible pairs in the subsequent measures. A power spectrogram with analysis window of size 46 ms was computed, leading to $F = 513$ frequency bins and $N = 676$ time frames. We ran MMLE with $K = 20$ on the spectrogram. We reconstructed STFT component estimates from the factorization $\hat{W}\hat{H}$, where $\hat{W}$ is the MMLE dictionary estimate and $\hat{H} = \langle H \rangle_q$. We used the minimum mean square error (MMSE) estimate given by $\hat{c}_{fkn} = g_{fkn} \cdot x_{fn}$, where $g_{fkn}$ is the time-frequency Wiener mask defined by $\hat{w}_{fk}\hat{h}_{kn}/\sum_j \hat{w}_{fj}\hat{h}_{jn}$. The estimated dictionary and the reconstructed components in the time domain after inverse STFT are shown in Fig. 3 (a). Out of the 20 components, 12 were assigned to zero during inference. The remaining 8 are displayed. 3 of the nonzero dictionary columns have very small values, leading to inaudible reconstructions. The five significant dictionary vectors correspond to the frequency templates of the four notes and the transients. For comparison, we applied the non-parametric approach by Hoffman *et al.* [11] on the same data with the same hyperparameters for $H$. The estimated dictionary and the reconstructed components are presented in Fig. 3 (b). 10 out of 20 components had very small weight values. The most significant 8 of the remaining components are presented in the figure. These components do not exactly correspond to individual notes and transients as they did with MMLE. The fourth note is mainly represented in the fifth component, but partially appears in the first three components as well. In general, the performance of the nonparametric approach depends more on initialization, i.e., requires more repetitions than MMLE. For the above results, we used 200 repetitions for the nonparametric method and 20 for MMLE (without annealing, same stopping criterion) and chose the repetition with the highest likelihood.

## 5.3 Decomposition of a real song

In this last experiment, we decompose the first 40 seconds of *God Only Knows* by the Beach Boys. This song was produced in mono and we retrieved a downsampled version of it at 22kHz from the CD release. We computed a power spectrogram with 46 ms analysis window and ran our VB algorithm with $K = 50$. Fig. 4 displays the original data, and two examples of estimated time-frequency masks and reconstructed components. The figure also shows the variance of the reconstructed components and the evolution of the variational bound along iterations. In this example, 5 components out of the 50 are completely pruned in the factorization and 7 others are inaudible. Such decomposition can be used in various music editing settings, for example for mono to stereo remixing, see, e.g., [3].

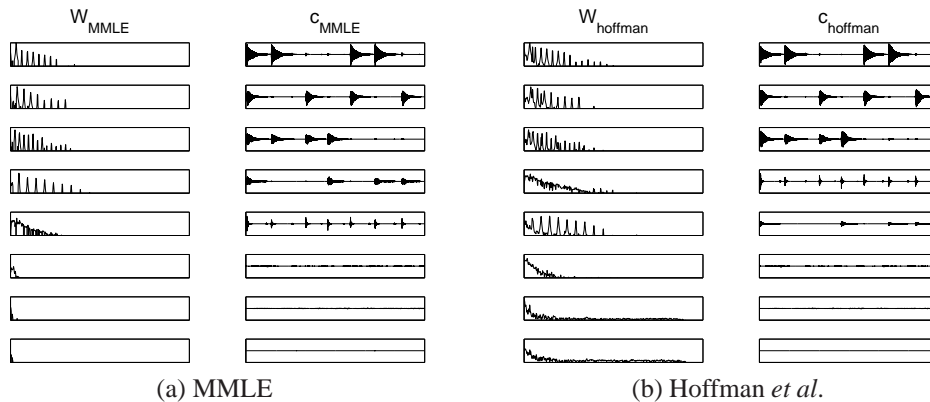

(a) MMLE                                      (b) Hoffman *et al.*

Figure 3: The estimated dictionary and the reconstructed components by MMLE and the nonparametric approach by Hoffman *et al.* with $K = 20$.

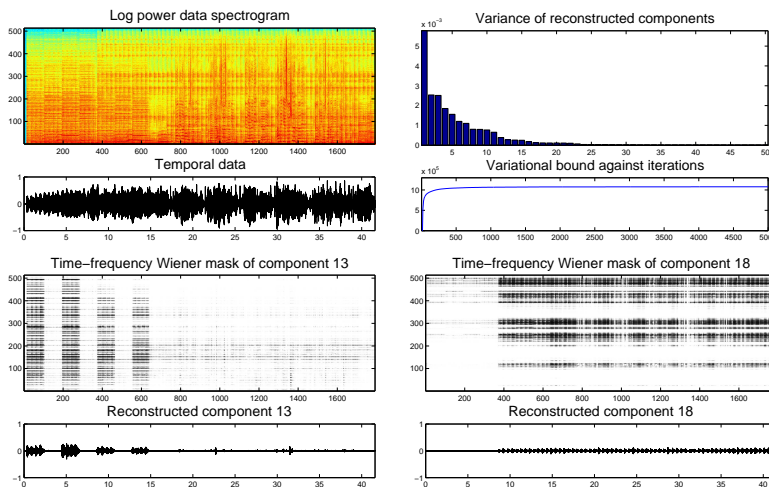

Figure 4: Decomposition results of a real song. The Wiener masks take values between 0 (white) and 1 (black). The first example of reconstructed component captures the first chord of the song, repeated 4 times in the intro. The other component captures the cymbal, which starts with the first verse of the song.

**Acknowledgments**

This work is supported by project ANR-09-JCJC-0073-01 TANGERINE (Theory and applications of nonnegative matrix factorization).

## 6  Conclusions

In this paper we have challenged the standard NMF approach to nonnegative dictionary learning, based on maximum joint likelihood estimation, with a better-posed approach consisting in maximum marginal likelihood estimation. The proposed algorithm based on variational inference has comparable computational complexity to standard NMF/MJLE. Our experiments on synthetical and real data have brought up a very attractive feature of MMLE, namely its self-ability to discard irrelevant columns in the dictionary, without resorting to elaborate schemes such as Bayesian nonparametrics.

**References**

[1] D. D. Lee and H. S. Seung. Learning the parts of objects with nonnegative matrix factorization. *Nature*, 401:788–791, 1999.

[2] C. Févotte and A. T. Cemgil. Nonnegative matrix factorisations as probabilistic inference in composite models. In *Proc. 17th European Signal Processing Conference (EUSIPCO)*, pages 1913–1917, Glasgow, Scotland, Aug. 2009.

[3] C. Févotte, N. Bertin, and J.-L. Durrieu. Nonnegative matrix factorization with the Itakura-Saito divergence. With application to music analysis. *Neural Computation*, 21(3):793–830, Mar. 2009.

[4] David M. Blei, Andrew Y. Ng, and Michael I. Jordan. Latent Dirichlet allocation. *Journal of Machine Learning Research*, 3:993–1022, Jan. 2003.

[5] Thomas Hofman. Probabilistic latent semantic indexing. In *Proc. 22nd International Conference on Research and Development in Information Retrieval (SIGIR)*, 1999.

[6] E. Gaussier and C. Goutte. Relation between PLSA and NMF and implications. In *Proc. 28th annual international ACM SIGIR conference on Research and development in information retrieval (SIGIR'05)*, pages 601–602, New York, NY, USA, 2005. ACM.

[7] M. Welling, C. Chemudugunta, and N. Sutter. Deterministic latent variable models and their pitfalls. In *SIAM Conference on Data Mining (SDM)*, pages 196–207, 2008.

[8] W. L. Buntine and A. Jakulin. Discrete component analysis. In *Lecture Notes in Computer Science*, volume 3940, pages 1–33. Springer, 2006.

[9] John F. Canny. GaP: A factor model for discrete data. In *Proceedings of the 27th ACM international Conference on Research and Development of Information Retrieval (SIGIR)*, pages 122–129, 2004.

[10] O. Dikmen and C. Févotte. Maximum marginal likelihood estimation for nonnegative dictionary learning. In *Proc. of International Conference on Acoustics, Speech and Signal Processing (ICASSP'11)*, Prague, Czech Republic, 2011.

[11] M. Hoffman, D. Blei, and P. Cook. Bayesian nonparametric matrix factorization for recorded music. In *Proc. 27th International Conference on Machine Learning (ICML)*, Haifa, Israel, 2010.

[12] D. R. Hunter and K. Lange. A tutorial on MM algorithms. *The American Statistician*, 58:30 – 37, 2004.

[13] Y. Cao, P. P. B. Eggermont, and S. Terebey. Cross Burg entropy maximization and its application to ringing suppression in image reconstruction. *IEEE Transactions on Image Processing*, 8(2):286–292, Feb. 1999.

[14] C. M. Bishop. *Pattern Recognition And Machine Learning*. Springer, 2008. ISBN-13: 978-0387310732.

[15] K. Katahira, K. Watanabe, and M. Okada. Deterministic annealing variant of variational Bayes method. In *International Workshop on Statistical-Mechanical Informatics 2007 (IWSMI 2007)*, 2007.

[16] D. Donoho and V. Stodden. When does non-negative matrix factorization give a correct decomposition into parts? In Sebastian Thrun, Lawrence Saul, and Bernhard Schölkopf, editors, *Advances in Neural Information Processing Systems 16*. MIT Press, Cambridge, MA, 2004.

[17] D. J. C. Mackay. Probable networks and plausible predictions – a review of practical Bayesian models for supervised neural networks. *Network: Computation in Neural Systems*, 6(3):469–505, 1995.

[18] C. M. Bishop. Bayesian PCA. In *Advances in Neural Information Processing Systems (NIPS)*, pages 382–388, 1999.

## Footnotes

[1] A complex random variable has distribution $\mathcal{N}_c(\mu, \lambda)$ if and only if its real and imaginary parts are independent and distributed as $\mathcal{N}(\Re(\mu), \lambda/2)$ and $\mathcal{N}(\Im(\mu), \lambda/2)$, respectively.

[2]MATLAB code is available at http://perso.telecom-paristech.fr/~dikmen/nips11/
